# Rapid Inference on a Novel AND/OR graph for Object Detection, Segmentation and Parsing

**Yuanhao Chen**
Department of Automation
University of Science and Technology of China
yhchen4@ustc.edu.cn

**Long (Leo) Zhu**
Department of Statistics
University of California, Los Angeles
lzhu@stat.ucla.edu

**Chenxi Lin**
Microsoft Research Asia
chenxil@microsoft.com

**Alan Yuille**
Department of Statistics, Psychology and Computer Science
University of California, Los Angeles
yuille@stat.ucla.edu

**Hongjiang Zhang**
Microsoft Advanced Technology Center
hjzhang@microsoft.com

## Abstract

In this paper we formulate a novel AND/OR graph representation capable of describing the different configurations of deformable articulated objects such as horses. The representation makes use of the *summarization principle* so that lower level nodes in the graph only pass on summary statistics to the higher level nodes. The probability distributions are invariant to position, orientation, and scale. We develop a novel inference algorithm that combined a bottom-up process for proposing configurations for horses together with a top-down process for refining and validating these proposals. The strategy of surround suppression is applied to ensure that the inference time is polynomial in the size of input data. The algorithm was applied to the tasks of detecting, segmenting and parsing horses. We demonstrate that the algorithm is fast and comparable with the state of the art approaches.

## 1  Introduction

Most problems in machine intelligence can be formulated as probabilistic inference using probabilistic models defined on structured knowledge representations. Important examples include stochastic grammars [11] and, in particular, AND/OR graphs [8],[4],[10]. In practice, the nature of the representations is constrained by the types of inference algorithms which are available. For example, probabilistic context free grammars for natural language processing have a natural one-dimensional structure which makes it practical to use dynamic programming (DP) for inference [11]. But DP can not be directly applied to vision problems which lack this one-dimensional structure.

In this paper, we address the problem of detecting, segmenting and parsing articulated deformable objects, such as horses, in cluttered backgrounds. Formulating these tasks as statistical inference requires a representation that can deal with all the different possible configurations of the object including: (a) the different appearances of sub-configurations (e.g. the variable number of visible legs of a horse) and (b) the unknown location, size, and orientation of the object. In addition, we must specify a fast inference algorithm that can rapidly search over all the possible configurations of the object.

We first specify a novel AND/OR graph representation that efficiently allows for all the different configurations of an articulated deformable object (i.e. only a small number of nodes are required). The design of this graph uses the principle of *summarization*, so that lower level nodes in the graph only pass on summary statistics (abstract) to the higher level nodes. More precisely, the nodes of the AND/OR graph specify the position, orientation and scale of sub-configurations of the object (together with an index variable which specifies which sub-configurations of the object are present). The probability distribution defined on this representation obeys the Markov condition. It is designed to be invariant to the position, pose, and size of the object. In this paper, the representation and probability distributions are specified by hand.

We next describe an algorithm for performing inference over this representation. This is a challenging task since the space of possible configurations is enormous and there is no natural ordering to enable dynamic programming. Our algorithm combines a bottom-up process that makes proposals for the possible configurations of the object followed by a top-down process that refines and validates (or rejects) these proposals. The bottom-up process is based on the principle of *compositionality*, where we combine proposals for sub-configurations together to form proposals for bigger configurations. To avoid a combinational explosion of proposals, we prune out proposals in two ways: (i) removing proposals whose goodness of fit is poor, and (ii) performing *surround suppression* to represent *local clusters* of proposals by a single *max-proposal*. The top-down process refines and validates (or rejects) proposals for the entire configuration by allowing max-proposals to be replaced by other proposals from their local clusters if these leads to a better overall fit. In addition, the top-down process estimates the boundary of the object and performs segmentation. Surround suppression ensures that the computional complexity of the inference algorithm is polynomial in the size of image (input data).

The algorithm was tested for the task of detecting horses in cluttered backgrounds, using a standard dataset [2]. The input to the algorithm are the set of oriented edgelets detected in the image. The results show that the algorithm is very fast (approximately 13 seconds) for detecting, parsing, and segmenting the horses. Detection and segmentation are tested on 328 images and we obtain very good results using performance measures compared to ground truth. Parsing is tested on 100 images and we also obtain very good performance results (there are fewer test images for this task because it is harder to obtain datasets with ground truth parsing).

## 2  Background

Detection, segmentation and parsing are all challenging problems. Most computer vision systems only address one of these tasks. There has been influential work on detection [6], [9] and on the related problem of registration [5],[1]. Work on segmentation includes [12], [13], [3], [7], [14], [18], [17] and [16]. Much of this work is formulated, or can be reformulated, in terms of probabilistic inference. But the representations are fixed graph structures defined at a single scale. This restricted choice of representation enables the use of standard inference algorithms (e.g. the hungarian algorithm, belief propagation) but it puts limitations on the types of tasks that can be addressed (e.g. it makes parsing impossible), the number of different object configurations that can be addressed, and on the overall performance of the systems.

In the broader context of machine learning, there has been a growing use of probabilistic models defined over variable graph structures. Important examples include stochastic grammars which are particularly effective for natural language processing [11]. In particular, vision researchers have advocated the use of probability models defined over AND/OR graphs [4],[10] where the OR nodes enable the graph to have multiple structures. Similar AND/OR graphs have been used in other machine learning problems [8].

But the representational power of AND/OR graphs comes at the price of increased computational demand for performing inference (or learning). For one dimensional problems, such as natural language processing, this can be handled by dynamic programming. But computation becomes considerably harder for vision problems and it is not clear how to efficiently search over the large number of configurations of an AND/OR graph. The inference problem simplifies significantly if the OR nodes are restricted to lie at certain levels of the graph (e.g. [15], [20]), but these simplifications are not suited to the problem we are addressing.

# 3 The AND/OR Graph Representation

## 3.1 The topological structure of the AND/OR graph

The structure of an AND/OR graph is represented by a graph $G = (V, E)$ where $V$ and $E$ denote the set of vertices and edges respectively. The vertex set $V$ contains three types of nodes,"OR","AND" and "LEAF" nodes which are depicted in figure (1) by circles, rectangles and triangles respectively. These nodes have attributes including position, scale, and orientation. The edge set $E$ contains vertical edges defining the topological structure and horizontal edges defining spatial constraints on the node attributes. For each node $\nu \in V$, the set of its child nodes is defined by $T_\nu$.

The directed (vertical) edges connect nodes at successive levels of the tree. They connect: (a) the AND nodes to the OR nodes, (b) the OR nodes to the AND nodes, and (c) the AND nodes to the LEAF nodes. The LEAF nodes correspond directly to points in the image. Connection types (a) and (c) have fixed parent-child relationships, but type (b) has switchable parent-child relationship (i.e. the parent is connected to only one of its children, and this connection can switch). The horizontal edges only appear relating the children of the AND nodes. They correspond to Markov Random Fields (MRF's) and define spatial constraints on the node attributes. These constraints are defined to be invariant to translation, rotation, and scaling of the attributes of the children.

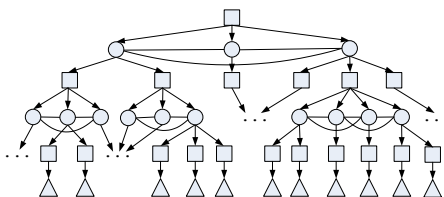

Figure 1: The AND/OR representation of the object.

The AND/OR graph we use in this paper is represented more visually in figure (2). The top node shows all the possible configurations of the horse (there are 40 in this paper). These configurations are obtained by AND-ing sub-configurations corresponding to the head, back, lower torso, and back legs of the horse (see circular nodes in the second row). Each of these sub-configurations has different *aspects* as illustrated by the AND nodes (rectangles in the third row). These sub-configurations, in turn, are composed by AND-ing more elementary configurations (see fourth row) which can have different aspects (see fifth row). (The topological structure of this representation is specified by the authors. Future work will attempt to learn it from examples).

## 3.2 The state variables defined on the AND/OR graph

A configuration of the AND/OR graph is an assignment of state variables $z = \{z_\nu\}$ with $z_\nu = (x_\nu, y_\nu, \theta_\nu, s_\nu, t_\nu)$ to each node $\nu$, where $(x, y)$, $\theta$ and $s$ denote image position, orientation, and scale respectively. The $t = \{t_\nu\}$ variable defines the specific topology of the graph and $t_\nu \in T_\nu$ . More precisely, $t_\nu$ defines the vertical parent-child relations by indexing the children of node $\nu$. $t_\nu$ is fixed and $t_\nu = T_\nu$ if $\nu$ is an AND node (because the node is always connected to all its children),

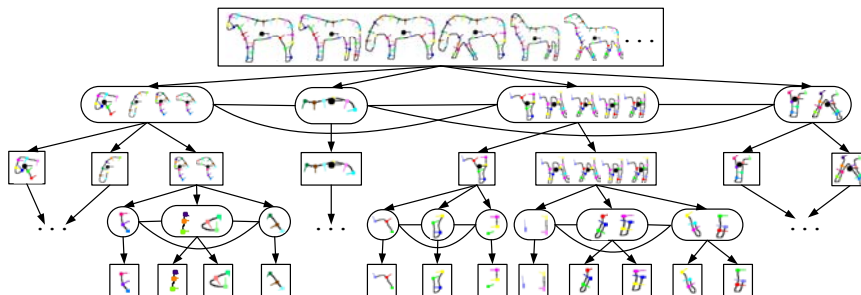

Figure 2: The AND/OR graph is an efficient way to representation different appearances of an object. The bottom level of the graph indicates points in the image. The higher levels indicating combinations of elementary configurations. The graph that we used contains eight levels (three lower levels are not depicted here due to lack of space).

but $t_\nu$ is a variable for an OR node $\nu$ (to enable sub-configurations to switch their appearances), see figure (2). We use the notation $Z_\nu$ to denote the state $z_\nu$ at node $\nu$, together with the states of all the descendent nodes of $\nu$ (i.e. the children of $\nu$, their children, and so on). The input to the graph is the data $d = \{d_\nu\}$ defined on the image lattice (at the lowest level of the hierarchy).

We define $V^{LEAF}(t)$, $V^{AND}(t)$, $V^{OR}(t)$ to be the set of LEAF, AND, and OR nodes which are active for a specific choice of the topology $t$. These sets can be computed recursively from the root node, see figure (2). The AND nodes in the second row (i.e. the second highest level of the graph) are always activated and so are the OR nodes in the third row. The AND nodes activated in the fourth row, and their OR node children in the fifth row, are specified by the $t$ variables assigned to their parent OR nodes. This process repeats till we reach the lowest level of the graph.

A novel feature of this AND/OR representation is that the node variables are the same at all levels of the hierarchy. We call this the *summarization principle*. It means that the state of an AND node will be a simple deterministic function of the state variables of the children (see section (3.3)). This differs from other AND/OR graphs [4],[10] where the node variables at different levels of the graph may be at different levels of abstraction. The use of the summarization principle enables us to define a successful inference algorithm.

## 3.3 The probability distribution for the AND/OR graph

The joint distribution on the states and the data is given by:

$$P(z,d) = \frac{1}{Z} \exp\{-E(d,z) - E_h(z) - E_v(z))\}. \tag{1}$$

where $d$ is the input data and $Z$ is the partition function.

The data term $E(d,z)$ is given by:

$$E(d,z) = \sum_{\nu \in V^{LEAF}(t)} f(d_\nu, z_\nu), \tag{2}$$

where $V^{LEAF}(t)$ is the set of the LEAF nodes and $f(.,.)$ is (negative) logarithm of Gaussian defined over grey-scale intensity gradient (i.e. magnitude and orientation). It encourages large intensity gradients in the image at locations of the nodes with the orientation roughly aligned to the orientation of the boundary.

The next two terms make use of the hierarchical structure. The horizontal component of the hierarchical shape prior is used to impose the horizontal connections at a range of scales and defined by

$$E_h(z) = \sum_{\nu \in V^{AND}(t)} \sum_{(\mu,\rho,\tau) \in t_\nu} g(z_\mu, z_\rho, z_\tau), \tag{3}$$

where $V^{AND}(t)$ is the set of AND nodes whose children are OR nodes and $g(z_\mu, z_\rho, z_\tau)$ is a (negative) logarithm of Gaussian distribution defined on the *invariant shape vector* $l(z_\mu, z_\rho, z_\tau)$ constructed from triple of childs nodes $(z_\mu, z_\rho, z_\tau)$ [20]. (This shape vector depends only on variables of the triple, such as the internal angles, that are invariant to the translation, rotation, and scaling of the triple. This ensures that the full probability distribution is also invariant to these transformations). The summation is over all triples formed by the child nodes of each parent, see figures (2). (Each node has at most four children, which restricts the set of triplets). The parameters of the Gaussian are fixed.

The vertical component $E_v(z)$ is used to hold the structure together by relating the state of the parent nodes to the state of its children. $E_v(z)$ is divided into three vertical energy terms denoted by $E_v^a(z)$, $E_v^b(z)$ and $E_v^c(z)$ which correspond to type(a), type(b) and type(c) vertical connections respectively. Hence we have

$$E_v(z) = E_v^a(z) + E_v^b(z) + E_v^c(z) \tag{4}$$

$E_v^a(z)$ specifies the coupling from the AND node to the OR node. The state of the parent node is determined precisely by the states of the child nodes. This is defined by:

$$E_v^a(z) = \sum_{\nu \in V^{AND}(t)} h(z_\nu; \{z_\mu \text{ s.t.} \mu \in t_\nu\}), \tag{5}$$

where $h(.,.) = 0$ if the average orientations and positions of the child nodes are equal to the orientation and position of the parent node (i.e. the vertical constraints are "hard"). If they are not consistent, then $h(.,.) = \kappa$, where $\kappa$ is a large positive number.

$E_v^b(z)$ accounts for the probability of the assignments of the connections from OR nodes to AND nodes.

$$E_v^b(z) = \sum_{\nu \in V^{OR}(t)} \lambda_\nu(t_\nu), \qquad (6)$$

where $\lambda_\nu()$ is the potential function which encodes the weights of the assignments determined by $t_\nu$.

The energy term $E_v^c(z)$ defines the connection from the lowest AND nodes to the LEAF nodes. This is similar to the definition of $E_v^a(z)$, and $E_v^c(z)$ is given by:

$$E_v^c(z) = \sum_{t_\nu \in V^{LEAF}(t)} h(z_\nu; z_{t_\nu}), \qquad (7)$$

where $h(.,.) = 0$ if the orientation and position of the child (LEAF) node is equal to the orientation and position of the parent (AND) node. If they are not consistent, then $h(.,.) = \kappa$.

Finally, we can compute the energy of the sub-tree for a particular node $\nu$ as root node. The sub-tree energy is useful when performing inference, see section (4). This is computed by summation over all the potential functions associating to the node $\nu$ and its descendants. This energy is defined by:

$$E_\nu(Z_\nu) = E(d, z) + E_h(z) + E_v(z). \qquad (8)$$

where $z \in Z_\nu$ and $V^{LEAF}(t), V^{AND}(t), V^{OR}(t)$ in the summation of each term are defined in the set of the node $\nu$ and its descendants.

Now we have specified a complete probability distribution for the graph. But this model is unable to do segmentation (since it has a limited number of nodes at the lowest level). To obtain a closed boundary based on the states of the leaf nodes, an extra energy term $E_0(d, z)$ at level $l = 0$ must be added to the exponent in equation (1). $E_0(d, z)$ is constructed similarly to that of Coughlan *et al* [6]. It is of form:

$$E_0(z) = \sum_{\nu \in V^{LEAF}} \sum_{\rho \in C(\nu, \nu')} \{f(d_\rho, z_\rho) + g(z_\rho, z_{\rho'})\}, \qquad (9)$$

where $\nu$ and $\nu'$ are neighbors at level 1, $C(\nu, \nu')$ is a curve connecting $\nu$ to $\nu'$ containing a fixed number of points, and $\rho'$ is the neighbor of $\rho$. The function $g(.,.)$ takes the (negative) logarithm of Gaussian form to define the prior on the orientation and scale. This energy term ensures that the leaf nodes are connected by a closed boundary which is used for segmentation.

## 4   Inference: Bottom-up and Top-down Processing

The task of the inference algorithm is to find a maximum a posteriori estimate of the state variables $z$:

$$z^* = \arg\max p(z|d) = \arg\max p(d|z)p(z), \qquad (10)$$

where $p(d|z)p(z) = p(d, z)$ is defined in equation (1).

The inference algorithm (see the pseudo code in figure (3)) contains a compositional bottom-up stage which makes proposals for the node variables in the tree. This is followed by a top-down stage which refines and validates the proposals. We use the following notation. Each node $\nu^l$ at level $l$ has a set of *proposals* $\{P_{\nu,a}^l\}$ where $a$ indexes the proposals (see table (2) for the typical number of proposals). There are also max-proposals $\{MP_{\nu,a}^l\}$, indexed by $a$, each associated with a local cluster $\{CL_{\nu,a}^l\}$ of proposals (see table (2) for the typical number of max-proposals). Each proposal, or max-proposal, is described by a state vector $\{z_{\nu,a}^l : a = 1, ..., M_\nu^l\}$, the state vectors for it and its descendants $\{Z_{\nu,a}^l : a = 1, ..., M_\nu^l\}$, and an energy function score $\{E_\nu^l(Z_{\nu,a}^l) : a = 1, ..., M_\nu^l\}$.

We obtain the proposals by a bottom-up strategy starting at level $l = 2$ (AND node) of the tree. For a node $\nu^2$ we define windows $\{W_{\nu,a}^2\}$ in space, orientation, and scale. We exhaustively search for all configurations within this window which have a score (goodness of fit criterion) $E_\nu^2(P_{\nu,a}^2) < K_2$, where $K_2$ is a fixed threshold. For each window $W_{\nu,a}^2$, we select the configuration with smallest

- Bottom-Up($MP^1$)
  Loop : $l = 2$ to $L$, for each node $\nu$ at level l
    - IF $\nu$ is an OR node
      1. Union: $\{MP_{\nu,b}^l\} = \bigcup_{\rho \in T_\nu, a=1,\ldots,M_\rho^{l-1}} MP_{\rho,a}^{l-1}$
    - IF $\nu$ is an AND node
      1. Composition: $\{P_{\nu,b}^l\} = \oplus_{\rho \in T_\nu, a=1,\ldots,M_\rho^{l-1}} MP_{\rho,a}^{l-1}$
      2. Pruning: $\{P_{\nu,a}^l\} = \{P_{\nu,a}^l | E_\nu(P_{\nu,a}^l) < K_l\}$
      3. Surround Suppression: $\{(MP_{\nu,a}^l, CL_{\nu,a}^l)\} = SurroundSuppression(\{P_{\nu,a}^l\}, \epsilon_W)$ where $\epsilon_W$ is the size of the window $W_\nu^l$ defined in space, orientation, and scale.
- Top-Down($MP^L, CL^L$):
  $MP^* = \arg\min_{a=1,\ldots,M_\nu^L, \tilde{P}=MP_{\nu,a}^L} ChangeProposal(\tilde{P}, MP_{\nu,a}^L, CL_{\nu,a}^L)$
- $ChangeProposal(\tilde{P}, MP_{\nu,a}^l, CL_{\nu,a}^l)$
    - IF $\nu$ is an OR node
      1. $ChangeProposal(\tilde{P}, MP_{t_\nu,a}^{l-1}, CL_{t_\nu,a}^{l-1})$
    - IF $\nu$ is an AND node
      1. $\dot{P} = \tilde{P} \ominus MP_{\nu,a}^l$
      2. $\ddot{P} = \arg\min_{P_{\nu,a'}^l \in CL_{\nu,a}^l} E_\nu(P_{\nu,a'}^l \oplus \dot{P}) + E_0(P_{\nu,a'}^l \oplus \dot{P})$ ($E_0()$ is obtained by dynamic programming)
      3. $\tilde{P} = \dot{P} \oplus \ddot{P}$
      4. Loop: for each $\rho \in T_\nu, \rho \ni V^{LEAF}$ and $b$ s.t. $MP_{\rho,b}^{l-1} \in \ddot{P}$
        * $ChangeProposal(\tilde{P}, MP_{\rho,b}^{l-1}, CL_{\rho,b}^{l-1})$
    - Return $\tilde{P}$ and its score $E_\nu(\tilde{P}) + E_0(\tilde{P})$

**Figure 3:** Bottom-up and Top-down Processing. $\oplus$ denotes the operation of combining two proposals. $\ominus$ denotes the operation of removing a part from a proposal.

score to be the proposal $MP_{\nu,a}^2$ and store the remaining proposals below threshold in the associated cluster $CL_{\nu,a}^2$. This window enforces *surround suppression* which performs clustering to keep the proposal with the maximum score in any local window. Surround suppression grantees the number of the remaining proposals at each level is proportional to the size of image (input data). This strategy ensures that we do not obtain too many proposals in the hierarchy and avoid a combinatorial explosion of proposals. We will analyze this property empirically in section 6. The procedure is repeated as we go up the hierarchy. Each parent node $\nu^{l+1}$ produces proposals $\{P_{\nu,a}^{l+1}\}$, and associated clusters $\{CL_{\nu,a}^{l+1}\}$, by combining the proposals from its children. All proposals are required to have scores $E_\nu^{l+1}(Z_\nu^{l+1}) < K_{l+1}$, where $K_l$ is a threshold.

The bottom-up process provides us with a set of proposals at the root (top) node. These proposals give a set of state vectors for the hierarchy for all nodes down to level $l = 1$, $\{Z_{\nu_0,a}^L : a = 1,\ldots,M_0^L\}$, where $\nu_0$ denotes the root node. In the top-down processing, for each proposal $a$ at the root node, we fix the state vectors of $Z_{\nu_0,a}^L$ and obtain the state of the level $l = 0$ variables (on the image lattice) by minimizing $E_\nu(Z_{\nu_0,a}^L) + E_0(Z_{\nu_0,a}^L)$ which is performed by dynamic programming, with the constraint that the level $l = 1$ nodes are fixed and the boundary contour must pass through them. The output of dynamic programming is a dense boundary contour. Next we refine the solution for each proposal at the root node by recursively changing the parts of the proposal. This is performed using the clusters associated with the proposals at each node. Each element of the cluster is an alternative proposal for the state of that node. The use of these clusters enables us to perform a set of transformations which may give a lower-energy configuration. The basic moves are to change the state vector of a node in the tree hierarchy to another state in the same proposal cluster, and then to determine the zeroth level nodes – for the appropriate segment of the contour – by dynamic programming. Change to the state vectors at high levels of the hierarchy will cause large changes to the boundary contour. Changes at the lower levels will only cause small changes. The procedure is repeated as we examine each node in the hierarchy recursively.

# 5 Complexity of Representation and Inference

We now quantify the representational power of the AND/OR graph and the complexity of the inference algorithm. These complexity measures depend on the following quantities: (i) the number $M$ of AND nodes connecting to OR nodes, (ii) the maximum number $K$ of children of AND nodes (we restrict $K \leq 4$), (iii) the maximum number $W$ of children of OR nodes, (iv) the number $h$ of levels

Table 1: Performance for parsing, segmentation and detection. The table compares the results for the hierarchial model (without OR nodes) and AND/OR graph with two inference algorithms, i.e. (a) bottom-up only. (b) bottom-up and top-down.

| Model | Testing Size | Parsing | Segmentation | Detection | Time |
|---|---|---|---|---|---|
| Hierarchical Model (a) | 328 | 18.7 | 81.3% / 73.4% | 86.0 (282 / 328) | 3.1s |
| Hierarchical Model (b) | 328 | 17.6 | 83.3% / 74.2% | 88.4 (290 / 328) | 6.1s |
| And/Or Graph (a) | 328 | 13.2 | 81.3% / 74.5% | 84.5 (277 / 328) | 4.5s |
| And/Or Graph (b) | 328 | 12.5 | 87.1% / 75.8% | 91.2 (299 / 328) | 13.2s |
| [16] | 172 | - | 86.2% / 75.0% | - | - |

Table 2: Complexity Analysis.

| Level | Nodes | Aspects | Max-Proposals | Proposals | Time |
|---|---|---|---|---|---|
| 8 | 1 | 12 | 11.1 | 2058.8 | 1.206s |
| 6 | 8 | 1.5 | 30.6 | 268.9 | 1.338s |
| 4 | 27 | 1 | 285.1 | 1541.5 | 1.631s |
| 2 | 68 | 1 | 172.2 | 1180.7 | 0.351s |

containing OR nodes with more than one child node, (v) the number $S$ of clusters for AND nodes (recall that the cluster is defined over image position, orientation and scale). In the experiments reported in this paper we have $K = 4, W = 3, h = 2, M = 36$. The number of proposals is linearly proportional to the size of the image.

The representational power is given by the number of different topological configurations of the AND/OR graph. It is straightforward to show that this is bounded above by $W^{K^h}$. In our experiments, the number of different topological configurations is $40$. The complexity of our algorithm can also be bounded above by $M \times W^K \times S^K$. This shows that the algorithm speed is polynomial in $W$ and $S$ (and hence in the image size). The complexity for our experiments is reported in section (6).

# 6 Results

The experimental results are obtained on 328 horse images [2] for detection and segmentation. We use 100 images for parsing (which requires more work to get groundtruth). The AND/OR model has 40 possible configurations. Some typical parsed and segmentation results are shown in figure (4).

In table (1) we compare the performances between the AND/OR graph with 40 configurations and a simple hierarchical model with only one configuration (each OR node has one child node). Column 3 gives the parsing accuracy – the average error of the position of leaf node is 10 pixels. Column 4 gives the precision and recall at the pixel level respectively (i.e. is the pixel inside the object or not). Column 5 quantifies the detection. We rate detection as a success if the area of intersection of the detected object region and the true object region is greater than half the area of the union of these regions. The last column shows the average time taken for one image. The AND/OR graph outperforms the simple hierarchical model in all tasks with two times more cost. Hierarchical model is only capable of locating the main body while AND/OR graph catches more details like legs, heads under different poses. Compared to [16] where the training and evaluation are performed with half of the data set, our method (evaluated on the whole data set) achieves better performance of segmentation with simpler feature. (Their method is unable to do parsing and detection.)

Table (2) shows the complexity properties of the algorithm. We described the AND levels only (the model has 8 levels). The computation for the OR-nodes is almost instantaneous (you just need to list the proposals from all its children AND nodes) so we do not include it. Column 2 gives the number of nodes at each level. Column 3 states the average number of *aspects* [1] of the AND nodes at each level. Column 4 states the average number of max-proposals for each node. Column 5 gives the average number of proposals. Column 6 gives the time. Observe that the number of proposals increases by an order of magnitude from level 6 to level 8. This is mostly due to the similar increase in the number of aspects (the more the number of aspects, the more the number of proposals needed to cover them). But surround suppression is capable of reducing the number of proposals greatly (compare the numbers of Max-proposals and proposals in Table (2)).

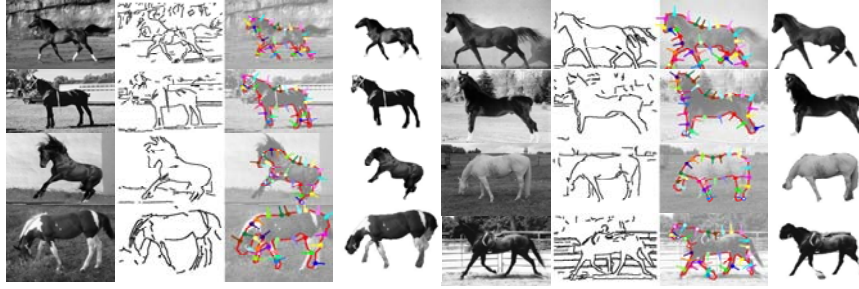

Figure 4: The parsed results. From left to right: original image, edge map, parsed result and segmentation. In the edge map, one can observe that some parts are missing or very ambiguous with low level cues. The colored dots correspond to the leaf nodes of the object.

# 7 Conclusion

We formulated a novel AND/OR graph representation capable of describing the different configurations of deformable articulated objects. The representation makes use of the summarization principle. We developed a novel inference algorithm that combined a bottom-up process for proposing configurations for horses together with a top-down process for refining and validating these proposals. Surround suppression ensures that the inference time is polynomial in the size of image. We demonstrated that the algorithm was fast and effective as evaluated by performance measures on a large dataset.

# 8 Acknowledgments

This research was supported by NSF grant 0413214 and the W.M. Keck foundation.

## Footnotes

[1]The definition of *aspects*. Let AND node $\nu$ have children OR nodes $\{\rho_i : i \in t_\nu\}$. This gives a set of grandchildren AND nodes $\bigcup_{i \in t_\nu} t_{\rho_i}$. The aspect of $\nu$ is $\prod_{i \in t_\nu} |t_{\rho_i}|$. The aspect of an AND node is an important concept. When passing up the proposals to an AND node we must take into account the number of aspects of this node. We can, in theory, have proposals for all possible aspects. The notion of aspects only goes down two levels.

# References

[1] S. Belongie, J. Malik, and J. Puzicha. Shape Matching and Object Recognition Using Shape Contexts. PAMI, 2002.

[2] E. Borenstein and S. Ullman. Class-specific, top-down segmentation. ECCV, 2002.

[3] E. Borenstein and J. Malik. Shape Guided object segmentation. CVPR 06

[4] H. Chen, Z.J. Xu, Z.Q. Liu, and S.C. Zhu. Composite Templates for Cloth Modeling and Sketching. CVPR, 2006.

[5] H. Chui and A. Rangarajan, A New Algorithm for Non-Rigid Point Matching. CVPR, 2000.

[6] J.M. Coughlan, and S. Ferreira. Finding Deformable Shapes using Loopy Belief Propagation. ECCV, 2002.

[7] T. Cour and J. Shi. Recognizing Objects by Piecing Together the Segmentation Puzzle. CVPR, 2007.

[8] H. Dechter and Robert Mateescu. AND/OR Search Spaces for Graphical Models. In Artificial Intelligence, 2006.

[9] R. Fergus, P. Perona and A. Zisserman. Object Class Recognition by Unsupervised Scale-Invariant Learning. CVPR, 2003.

[10] Y. Jin, S. Geman. Context and Hierarchy in a Probabilistic Image Model. CVPR 2006.

[11] D. Klein and C. Manning. Natural Language Grammar Induction Using a Constituent-Context Model. NIPS, 2001.

[12] M. P. Kumar, P. H. S. Torr and A. Zisserman. OBJ CUT, CVPR, 2005.

[13] B. Leibe, A. Leonardis and B. Schiele. Combined object categorization and segmentation with an implicit shape model. ECCV, 2004.

[14] A. Levin and Y. Weiss. Learning to Combine Bottom-up and Top-down Segmentation. ECCV, 2006

[15] M. Meila and M. I. Jordan. Mixture of Trees: Learning with mixtures of trees. Journal of Machine Learning Research, 1, 1-48, 2000.

[16] X. Ren, C. Fowlkes, and J. Malik, Cue integration in figure/ground labeling. NIPS, 2005.

[17] P. Srinivasan and J. Shi. Bottom-up Recognition and Parsing of the Human Body. CVPR, 2007.

[18] J. Winn and N. Jojic. LOCUS: Learning Object Classes with Unsupervised Segmentation. ICCV, 2005

[19] L. Zhu, and A. Yuille. A Hierarchical Compositional System for Rapid Object Detection. NIPS 2006.

[20] L. Zhu, Y. Chen, and A. Yuille. Unsupervised Learning of a Probabilistic Grammar for Object Detection and Parsing. NIPS 2007.

